# Asynchronous Dynamics of Continuous Time Neural Networks

**Xin Wang**
Computer Science Department
University of California at Los Angeles
Los Angeles, CA 90024

**Qingnan Li**
Department of Mathematics
University of Southern California
Los Angeles, CA 90089-1113

**Edward K. Blum**
Department of Mathematics
University of Southern California
Los Angeles, CA 90089-1113

## ABSTRACT

Motivated by mathematical modeling, analog implementation and distributed simulation of neural networks, we present a definition of *asynchronous dynamics* of general CT dynamical systems defined by ordinary differential equations, based on notions of local times and communication times. We provide some preliminary results on globally asymptotical convergence of asynchronous dynamics for contractive and monotone CT dynamical systems. When applying the results to neural networks, we obtain some conditions that ensure additive-type neural networks to be asynchronizable.

## 1 INTRODUCTION

Neural networks are massively distributed computing systems. A major issue in parallel and distributed computation is synchronization versus asynchronization (Bertsekas and Tsitsiklis, 1989). To fix our idea, we consider a much studied additive-type model (Cohen and Grossberg, 1983; Hopfield, 1984; Hirsch, 1989) of a continuous-time (CT) neural network of $n$ neurons, whose dynamics is governed by

$$\dot{x}_i(t) = -a_i x_i(t) + \sum_{j=1}^{n} w_{ij} \sigma_j(\mu_j x_j(t)) + I_i, \quad i = 1, 2, ..., n, \tag{1}$$

with neuron states $x_i(t)$ at time $t$, constant decay rates $a_i$, external inputs $I_i$, gains $\mu_j$, neuron activation functions $\sigma_j$ and synaptic connection weights $w_{ij}$. Simulation and implementation of idealized models of neural networks such as (1) on centralized computers not only limit the size of networks, but more importantly preclude exploiting the inherent massive parallelism in network computations. A truly faithful analog implementation or simulation of neural networks defined by (1) over a distributed network requires that neurons follow a global clock $t$, communicate timed states $x_j(t)$ to all others instantaneously and synchronize global dynamics precisely all the time (e.g., the same $x_j(t)$ should be used in evolution of all $x_i(t)$ at time $t$). Clearly, hardware and software realities make it very hard and sometimes impossible to fulfill these requirements; any mechanism used to enforce such synchronization may have an important effect on performance of the network. Moreover, absolutely insisting on synchronization contradicts the biological manifestation of inherent asynchrony caused by delays in nerve signal propagation, variability of neuron parameters such as refractory periods and adaptive neuron gains. On the other hand, introduction of asynchrony may change network dynamics, for example, from convergent to oscillatory. Therefore, validity of asynchronous dynamics of neural networks must be assessed in order to ensure desirable dynamics in a distributed environment.

Motivated by the above issues, we study *asynchronous dynamics* of general CT dynamical systems with neural networks in particular. Asynchronous dynamics has been thoroughly studied in the context of iterative maps or discrete-time (DT) dynamical systems; see, e.g., (Bertsekas and Tsitsiklis, 1989) and references therein. Among other results are that $P$-contractive maps on $\mathbf{R}^n$ (Baudet, 1978) and continuous maps on partially ordered sets (Wang and Parker, 1992) are asynchronizable, i.e., any asynchronous iterations of these maps will converge to the fixed points under synchronous (or parallel) iterations. The synchronization issue has also been addressed in the context of neural networks. In fact, the celebrated DT Hopfield model (Hopfield, 1982) adopts a special kind of asynchronous dynamics: only one randomly chosen neuron is allowed to update its state at each iterative step. The issue is also studied in (Barhen and Gulati, 1989) for CT neural networks. The approach there is, however, to convert the additive model (1) into a DT version through the Euler discretization and then to apply the existing result for contractive mappings in (Baudet, 1978) to ensure the discretized system to be asynchronizable. Overall, studies for asynchronous dynamics of CT dynamical systems are still lacking; there are even no reasonable definitions for what it means, at least to our knowledge.

In this paper, we continue our studies on relationships between CT and DT dynamical systems and neural networks (Wang and Blum, 1992; Wang, Blum and Li, 1993) and concentrate on their asynchronous dynamics. We first extend a concept of asynchronous dynamics of DT systems to CT systems, by identifying the distinction between synchronous and asynchronous dynamics as (i) presence or absence of a common global clock used to synchronize the dynamics of the different neurons and (ii) exclusion or inclusion of delay times in communication between neurons, and present some preliminary results for asynchronous dynamics of contractive and monotone CT systems.

## 2   MATHEMATICAL FORMULATION

To be general, we consider a CT dynamical system defined by an $n$-dimensional system of ordinary differential equations,

$$\dot{x}_i(t) = f_i(x_1(t), ..., x_n(t)), \quad i = 1, 2, ..., n, \tag{2}$$

where $f_i : \mathbf{R}^n \to \mathbf{R}$ are continuously differentiable and $x(t) \in \mathbf{R}^n$ for all $t$ in $\mathbf{R}_+$ (the set of all nonnegative real numbers). In contrast to the asynchronous dynamics given below, dynamics of this system will be called *synchronous*. An *asynchronous scheme* consists of two families of functions $c_i : \mathbf{R}_+ \to \mathbf{R}_+$ and $\tau_j^i : \mathbf{R}_+ \to \mathbf{R}_+$, $i, j = 1, ..., n$, satisfying the following constraints: for any $t \geq 0$,

(i) Initiation: $c_i(t) \geq 0$ and $\tau_j^i(t) \geq 0$;

(ii) Non-starvation: $c_i$'s are differentiable and $\dot{c}_i(t) > 0$;

(iii) Liveness: $\lim_{t \to \infty} c_i(t) = \infty$ and $\lim_{t \to \infty} \tau_j^i(t) = \infty$;

(iv) Accessibility: $\tau_j^i(t) \leq c_j(t)$.

Given an asynchronous scheme $(\{c_i\}, \{\tau_j^i\})$, the associated *asynchronous dynamics* of the system (2) is the solution of the following parametrized system:

$$\dot{x}_i(c_i(t)) = f_i(x_1(\tau_1^i(t)), ..., x_n(\tau_n^i(t))). \tag{3}$$

We shall call this system an *asynchronized* system of the original one (2).

The functions $c_i(t)$ should be viewed as respective "local" times (or clocks) of components $i$, as compared to the "global" time (or clock) $t$. As each component $i$ evolves its state according to its local time $c_i(t)$, no shared global time $t$ is needed explicitly; $t$ only occurs implicitly. The functions $\tau_j^i(t)$ should be considered as time instants at which corresponding values $x_j$ of components $j$ are used by component $i$; hence the differences $(c_j(t) - \tau_j^i(t)) \geq 0$ can be interpreted as delay times in communication between the components $j$ and $i$. Constraint (i) reflects the fact that we are interested in the system dynamics after some global time instance, say 0; constraint (ii) states that the functions $c_i$ are monotone increasing and hence the local times evolve only forward; constraint (iii) characterizes the liveness property of the components and communication channels between components; and, finally, constraint (iv) precludes the possibility that component $i$ accesses states $x_j$ ahead of the local times $c_j(t)$ of components $j$ which have not yet been generated.

Notice that, under the assumption on monotonicity of $c_i(t)$, the inverses $c_i^{-1}(t)$ exist and the asynchronized system (3) can be transformed into

$$\dot{y}_i(t) = \dot{c}_i(t) \ f_i(\bar{y}_1^i(t), \bar{y}_2^i(t), ..., \bar{y}_n^i(t)) \tag{4}$$

by letting $y_i(t) = x_i(c_i(t))$ and $\bar{y}_j^i(t) = x_j(\tau_j^i(t)) = y_j(c_j^{-1}(\tau_j^i(t)))$ for $i, j = 1, 2, ..., n$. The vector form of (4) can be given by

$$\dot{y} = C'F[\bar{Y}] \tag{5}$$

where $y(t) = [y_1(t), ..., y_n(t)]^\top$, $C' = diag(dc_1(t)/dt, ..., dc_n(t)/dt)$, $F = [f_1, ..., f_n]^\top$, $\bar{Y} = [\bar{y}_j^i]$ and

$$F[\bar{Y}] = \begin{bmatrix} f_1(\bar{y}_1^1(t), \bar{y}_2^1(t), ..., \bar{y}_n^1(t)) \\ f_2(\bar{y}_1^2(t), \bar{y}_2^2(t), ..., \bar{y}_n^2(t)) \\ \vdots \\ f_n(\bar{y}_1^n(t), \bar{y}_2^n(t), ..., \bar{y}_n^n(t)) \end{bmatrix}.$$

Notice that the complication in the way $F$ applies to $\bar{Y}$ simply means that every component $i$ will use possibly different "global" states $[\bar{y}_1^i(t), \bar{y}_2^i(t), ..., \bar{y}_n^i(t)]$. This peculiarity makes the equation (5) fit into none of the categories of general functional differential equations (Hale, 1977). However, if $\tau_j^i(t)$ for $i = 1, ..., n$ are equal, all the components will use a same global state $\bar{y} = [\bar{y}_1^1(t), \bar{y}_2^2(t), ..., \bar{y}_n^n(t)]$ and the asynchronized system (5) assumes a form of retarded functional differential equations,

$$\dot{y} = C'F(\bar{y}). \tag{6}$$

We shall call this case *uniformly-delayed*, which will be a main consideration in the next section where we discuss asynchronizable systems.

The system (5) includes some special cases. In a no communication delay situation, $\tau_j^i(t) = c_j(t)$ for all $i$ and the system (5) reduces to $\dot{y} = C'F(y)$. This includes the simplest case where the local times $c_i(t)$ are taken as constant-time scalings $c_i t$ of the global time $t$; specially, when all $c_i(t) = t$ the system goes back to the original one (2). If, on the other hand, all the local times are identical to the global time $t$ and the communication times take the form of $\tau_j^i(t) = t - \theta_j^i(t)$ one obtains a most general delayed system

$$\dot{y}_i(t) = f_i(y_1(t - \theta_1^i(t)), y_2(t - \theta_2^i(t)), ..., y_n(t - \theta_n^i(t))), \tag{7}$$

where the state $y_j(t)$ of component $j$ may have different delay times $\theta_j^i(t)$ for different other components $i$.

Finally, we should point out that the above definitions of asynchronous schemes and dynamics are analogues of their counterparts for DT dynamical systems (Bertsekas and Tsitsiklis, 1989; Blum, 1990). Usually, an asynchronous scheme for a DT system defined by a map $f : X \rightarrow X$, where $X = X_1 \times X_2 \times \cdots \times X_n$, consists of a family $\{T^i \subseteq \mathsf{N} \,|\, i = 1, ..., n\}$ of subsets of discrete times ($\mathsf{N}$) at which components $i$ update their states and a family $\{\tau_j^i : \mathsf{N} \rightarrow \mathsf{N} \,|\, i = 1, 2, ..., n\}$ of communication times. Asynchronous dynamics (or chaotic iteration, relaxation) is then given by

$$x_i(t+1) = \begin{cases} f_i(x_1(\tau_1^i(t)), ..., x_n(\tau_n^i(t))) & \text{if } t \in T^i \\ x_i(t) & \text{otherwise.} \end{cases}$$

Notice that the sets $T^i$ can be interpreted as local times of components $i$. In fact, one can define local time functions $c_i : \mathsf{N} \rightarrow \mathsf{N}$ as $c_i(0) = 0$ and $c_i(t+1) = c_i(t) + 1$ if $t \in T_i$ and $c_i(t)$ otherwise. The asynchronous dynamics can then be defined by

$$x_i(t+1) - x_i(t) = (c_i(t+1) - c_i(t))(f_i(x_1(\tau_1^i(t)), ..., x_n(\tau_n^i(t))) - x_i(t)),$$

which is analogous to the definition given in (4).

# 3   ASYNCHRONIZABLE SYSTEMS

In general, we consider a CT dynamical system as *asynchronizable* if its synchronous dynamics (limit sets and their asymptotic stability) persists for some set of asynchronous schemes. In many cases, asynchronous dynamics of an arbitrary CT system will be different from its synchronous dynamics, especially when delay times in communication are present. An example can be given for the network (1) with symmetric matrix $W$. It is well-known that (synchronous) dynamics of such networks is *quasi-convergent*, namely, all trajectories approach a set of fixed points (Hirsch, 1989). But when delay times are taken into consideration, the networks may have sustained oscillation when the delays exceed some threshold (Marcus and Westervelt, 1989). A more careful analysis on oscillation induced by delays is given in (Wu, 1993) for the networks with symmetric circulant weight matrices.

Here, we focus on asynchronizable systems. We consider CT dynamical systems on $\mathbf{R}^n$ of the following general form

$$A\dot{x}(t) = -x(t) + F(x(t)) \tag{8}$$

where $x(t) \in \mathbf{R}^n$, $A = diag(a_1, a_2, ..., a_n)$ with $a_i > 0$ and $F = [f_i] \in C^1(\mathbf{R}^n)$. It is easy to see that a point $x \in \mathbf{R}^n$ is a fixed point of (8) if and only if $x$ is a fixed point of the map $F$. Without loss of generality, we assume that 0 is a fixed point of the map $F$. According to (5), the asynchronized version of (8) for an arbitrary asynchronous scheme $(\{c_i\}, \{\tau_j^i\})$ is

$$A\dot{y} = C'(-\bar{y} + F[\bar{Y}]), \tag{9}$$

where $\bar{y} = [\bar{y}_1^1(t), \bar{y}_2^2(t), ..., \bar{y}_n^n(t)]$.

## 3.1   Contractive Systems

Our first effort attempts to obtain a result similar to the one for $P$-contractive maps in (Baudet, 1978). We call the system (8) *strongly $P$-contractive* if there is a symmetric and invertible matrix $S$ such that $|S^{-1}F(Sx)| \le |x|$ for all $x \in \mathbf{R}^n$ and $|S^{-1}F(Sx)| = |x|$ only for $x = 0$; here $|x|$ denotes the vector with components $|x_i|$ and $\le$ is component-wise.

**Theorem 1** *If the system (8) is strongly P-contractive, then it is asynchronizable for any asynchronous schemes without self time delays (i.e., $\tau_i^i(t) = c_i(t)$ for all $i = 1, 2, ..., n$).*

Proof. It is not hard to see that synchronous dynamics of a strongly $P$-contractive system is globally convergent to the fixed point 0. Now, consider the transformation $z = A^{-1}y$ and the system for $z$

$$A\dot{z} = C'(-z + S^{-1}F[S\bar{Z}]) = C'(-z + G[\bar{Z}]),$$

where $G[\bar{Z}] = S^{-1}FS[\bar{Z}]$. This system has the same type of dynamics as (9). Define a function $E : \mathbf{R}_+ \times \mathbf{R}^n \to \mathbf{R}_+$ by $E(t) = z^\top(t)Az(t)/2$, whose derivative with respect to $t$ is

$$\dot{E} = z^\top C' \left(-z + G(\bar{Z})\right) \le \|C'\| \left(-z^\top z + |z|^\top |G(\bar{Z})|\right) < \|C'\|(-z^\top z + |z|^\top |z|) \le 0.$$

Hence $E$ is an energy function and the asynchronous dynamics converges to the fixed point 0. □

Our second result is for asynchronous dynamics of contractive systems with no communication delay. The system (8) is called *contractive* if there is a real constant $0 \leq \alpha < 1$ such that

$$\|F(x) - F(y)\| \leq \alpha \|x - y\|$$

for all $x, y \in \mathbf{R}^n$; here $\| \cdot \|$ denotes the usual Euclidean norm on $\mathbf{R}^n$.

**Theorem 2** *If the system (8) is contractive, then it is asynchronizable for asynchronous schemes with no communication delay.*

Proof. The synchronous dynamics of contractive systems is known to be globally convergent to a unique fixed point (Kelly, 1990). For an asynchronous scheme with no communication delay, the system (8) is simplified to $A\dot{y} = C'(-y + F(y))$. We consider again the function $E = y^{\mathsf{T}} A y / 2$, which is an energy function as shown below.

$$\dot{E} = y^{\mathsf{T}} C'(-y + F(y)) \leq \|C'\| (-\|y\|^2 + \|y\| \|F(y)\|) < 0.$$

Therefore, the asynchronous dynamics converges to the fixed point 0. □

For the additive-type neural networks (1), we have

**Corollary 1** *Let the network (1) have neuron activation functions $\sigma_i$ of sigmoidal type with $0 < \sigma_i'(z) \leq \sup_{z \in \mathbf{R}} \sigma_i'(z) = 1$. If it satisfies the condition*

$$\|A^{-1} W M\| < 1, \tag{10}$$

*where $M = diag(\mu_1, ..., \mu_n)$, then it is asynchronizable for any asynchronous schemes with no communication delay.*

Proof. The condition (10) ensures the map $F(x) = A^{-1} W \sigma(M x) + A^{-1} I$ to be contractive. □

Notice that the condition (10) is equivalent to many existing ones on globally asymptotical stability based on various norms of matrix $W$, especially the contraction condition given in (Kelly, 1990) and some very recent ones in (Matsuoka, 1992). The condition (10) is also related very closely to the condition in (Barhen and Gulati, 1989) for asynchronous dynamics of a discretized version of (1) and the condition in (Marcus and Westervelt, 1989) for the networks with delay.

We should emphasize that the results in Theorem 2 and Corollary 1 do not directly follow from the result in (Kelly, 1990); this is because local times $c_i(t)$ are allowed to be much more general functions than linear ones $c_i t$.

## 3.2   Monotone Systems

A binary relation $\preceq$ on $\mathbf{R}^n$ is called a *partial order* if it satisfies that, for all $x, y, z \in \mathbf{R}^n$, (i) $x \preceq x$; (ii) $x \preceq y$ and $y \preceq x$ imply $x = y$; and (iii) $x \preceq y$ and $y \preceq z$ imply $x \preceq z$. For a partial order $\preceq$ on $\mathbf{R}^n$, define $\ll$ on $\mathbf{R}^n$ by $x \ll y$ iff $x \leq y$ and $x_i \neq y_i$ for all $i = 1, \cdots, n$. A map $F : \mathbf{R}^n \to \mathbf{R}^n$ is *monotone* if $x \preceq y$ implies

$F(x) \preceq F(y)$. A CT dynamical system of the form (2) is *monotone* if $x_1 \preceq x_2$ implies the trajectories $x_1(t), x_2(t)$ with $x_1(0) = x_1$ and $x_2(0) = x_2$ satisfy $x_1(t) \preceq x_2(t)$ for all $t \geq 0$ (Hirsch, 1988).

**Theorem 3** *If the map $F$ in (8) is monotone, then the system (8) is asynchronizable for uniformly-delayed asynchronous schemes, provided that all orbits $x(t)$ have compact orbit closure and there is a $t_0 > 0$ with $x(t_0) \gg x(0)$ or $x(t_0) \ll x(0)$.*

Proof. This is an application of a Henry's theorem (see Hirsch, 1988) that implies that the asynchronized system (9) in the no communication delay situation is monotone and Hirsch's theorem (Hirsch, 1988) that guarantees the asymptotic convergence of monotone systems to fixed points.                     □

**Corollary 2** *If the additive-type neural network (1) with sigmoidal activation functions is* cooperative *(i.e., $w_{ij} \geq 0$ for $i \neq j$ (Hirsch, 1988 and 1989)), then it is asynchronizable for uniformly-delayed asynchronous schemes, provided that there is a $t_0 > 0$ with $x(t_0) \gg x(0)$ or $x(t_0) \ll x(0)$.*

Proof. According to (Hirsch, 1988), cooperative systems are monotone. As the network has only bounded dynamics, the result follows from the above theorem. □

## 4    CONCLUSION

By incorporating the concepts of local times and communication times, we have provided a mathematical formulation of asynchronous dynamics of continuous-time dynamical systems. Asynchronized systems in the most general form haven't been studied in theories of dynamical systems and functional differential equations. For contractive and monotone systems, we have shown that for some asynchronous schemes, the systems are asynchronizable, namely, their asynchronizations preserve convergent dynamics of the original (synchronous) systems. When applying these results to the additive-type neural networks, we have obtained some special conditions for the networks to be asynchronizable.

We are currently investigating more general results for asynchronizable dynamical systems, with a main interest in oscillatory dynamics.

### References

G. M. Baudet (1978). Asynchronous iterative methods for multiprocessors. *Journal of the Association for Computing Machinery*, 25:226–244.

J. Barhen and S. Gulati (1989). "Chaotic relaxation" in concurrently asynchronous neurodynamics. In *Proceedings of International Conference on Neural Networks*, volume I, pages 619–626, San Diego, California.

Bertsekas and Tsitsiklis (1989). *Parallel and Distributed Computation: Numerical Methods*. Englewood Cliffs, NJ: Prentice Hall.

E. K. Blum (1990). Mathematical aspects of outer-product asynchronous content-addressable memories. *Biological Cybernetics*, 62:337–348, 1990.

E. K. Blum and X. Wang (1992). Stability of fixed-points and periodic orbits, and bifurcations in analog neural networks. *Neural Networks*, 5:577-587.

J. Hale (1977). *Theory of Functional Differential Equations*. New York: Springer-Verlag.

M. W. Hirsch (1988). Stability and convergence in strongly monotone dynamical systems. *J. reine angew. Math.*, 383:1–53.

M. W. Hirsch (1989). Convergent activation dynamics in continuous time networks. *Neural Networks*, 2:331–349.

J. Hopfield (1982). Neural networks and physical systems with emergent computational abilities. *Proc. Nat. Acad. Sci. USA*, 79:2554–2558.

J. Hopfield (1984). Neurons with graded response have collective computational properties like those of two-state neurons. *Proc. Nat. Acad. Sci. USA*, 81:3088-3092.

D. G. Kelly (1990). Stability in contractive nonlinear neural networks. *IEEE Trans. Biomedi. Eng.*, 37:231–242.

Q. Li (1993). Mathematical and Numerical Analysis of Biological Neural Networks. *Unpublished Ph.D. Thesis, Mathematics Department, University of Southern California*.

C. M. Marcus and R. M. Westervelt (1989). Stability of analog neural networks with delay. *Physical Review A*, 39(1):347–359.

K. Matsuoka (1992). Stability conditions for nonlinear continuous neural networks with asymmetric connection weights. *Neural Networks*, 5:495–500.

J. M. Ortega and W. C. Rheinboldt (1970). *Iterative solution of nonlinear equations in several variables*. New York: Academic Press.

X. Wang, E. K. Blum, and Q. Li (1993). Consistency on Local Dynamics and Bifurcation of Continuous-Time Dynamical Systems and Their Discretizations. To appear in *the AMS proceedings of Symposia in Applied Mathematics, Mathematics of Computation 1943 - 1993, Vancouver, BC, August, 1993, edited by W. Gautschi*.

X. Wang and E. K. Blum (1992). Discrete-time versus continuous-time neural networks. *Computer and System Sciences*, 49:1–17.

X. Wang and D. S. Parker (1992). Computing least fixed points by asynchronous iterations and random iterations. *Technical Report CSD-920025*, Computer Science Department, UCLA.

J.-H. Wu (1993). Delay-Induced Discrete Waves of Large Amplitudes in Neural Networks with Circulant Connection Matrices. *Preprint*, Department of Mathematics and Statistics, York University.
